# A POMDP Extension with Belief-dependent Rewards

**Mauricio Araya-López**        **Olivier Buffet**

**Vincent Thomas**        **François Charpillet**

Nancy Université / INRIA
LORIA – Campus Scientifique – BP 239
54506 Vandoeuvre-lès-Nancy Cedex – France
`firstname.lastname@loria.fr`

## Abstract

Partially Observable Markov Decision Processes (POMDPs) model sequential decision-making problems under uncertainty and partial observability. Unfortunately, some problems cannot be modeled with state-dependent reward functions, e.g., problems whose objective explicitly implies reducing the uncertainty on the state. To that end, we introduce $\rho$POMDPs, an extension of POMDPs where the reward function $\rho$ depends on the belief state. We show that, under the common assumption that $\rho$ is convex, the value function is also convex, what makes it possible to (1) approximate $\rho$ arbitrarily well with a piecewise linear and convex (PWLC) function, and (2) use state-of-the-art exact or approximate solving algorithms with limited changes.

## 1 Introduction

Sequential decision-making problems under uncertainty and partial observability are typically modeled using Partially Observable Markov Decision Processes (POMDPs) [1], where the objective is to decide how to act so that the sequence of visited states optimizes some performance criterion. However, this formalism is not expressive enough to model problems with any kind of objective functions.

Let us consider *active sensing* problems, where the objective is to act so as to acquire knowledge about certain state variables. Medical diagnosis for example is about asking the good questions and performing the appropriate exams so as to diagnose a patient at a low cost and with high certainty. This can be formalized as a POMDP by rewarding—if successful—a final action consisting in expressing the diagnoser's "best guess". Actually, a large body of work formalizes active sensing with POMDPs [2, 3, 4].

An issue is that, in some problems, the objective needs to be directly expressed in terms of the uncertainty/information on the state, e.g., to minimize the entropy over a given state variable. In such cases, POMDPs are not appropriate because the reward function depends on the state and the action, not on the knowledge of the agent. Instead, we need a model where the instant reward depends on the current *belief state*. The belief MDP formalism provides the needed expressiveness for these problems. Yet, there is not much research on specific algorithms to solve them, so they are usually forced to fit in the POMDP framework, which means changing the original problem definition. One can argue that acquiring information is always a means, not an end, and thus, a "well-defined" sequential-decision making problem with partial observability must always be modeled as a normal POMDP. However, in a number of cases the problem designer has decided to separate the task of looking for information from that of exploiting information. Let us mention two examples: (i) the

surveillance [5] and (ii) the exploration [2] of a given area, in both cases when one does not know what to expect from these tasks—and thus how to react to the discoveries.

After reviewing some background knowledge on POMDPs in Section 2, Section 3 introduces $\rho$POMDPs—an extension of POMDPs where the reward is a (typically convex) function of the belief state—and proves that the convexity of the value function is preserved. Then we show how classical solving algorithms can be adapted depending whether the reward function is piecewise linear (Sec. 3.3) or not (Sec. 4).

## 2 Partially Observable MDPs

The general problem that POMDPs address is for the agent to find a decision *policy* $\pi$ choosing, at each time step, the best action based on its past observations and actions in order to maximize its future gain (which can be measured for example through the total accumulated reward or the average reward per time step). Compared to classical deterministic planning, the agent has to face the difficulty to account for a system not only with uncertain dynamics but also whose current state is imperfectly known.

### 2.1 POMDP Description

Formally, POMDPs are defined by a tuple $\langle \mathcal{S}, \mathcal{A}, \Omega, T, O, r, b_0 \rangle$ where, at any time step, the system being in some state $s \in \mathcal{S}$ (the *state space*), the agent performs an action $a \in \mathcal{A}$ (the *action space*) that results in (1) a transition to a state $s'$ according to the *transition function* $T(s, a, s') = Pr(s'|s, a)$, (2) an observation $o \in \Omega$ (the *observation space*) according to the *observation function* $O(s', a, o) = Pr(o|s', a)$, and (3) a scalar *reward* $r(s, a)$. $b_0$ is the initial probability distribution over states. Unless stated otherwise, the state, action and observation sets are finite [6].

The agent can typically reason about the state of the system by computing a *belief state* $b \in \Delta = \Pi(\mathcal{S})$ (the set of probability distributions over $\mathcal{S}$),[1] using the following update formula (based on the Bayes rule) when performing action $a$ and observing $o$:

$$b^{a,o}(s') = \frac{O(s', a, o)}{Pr(o|a, b)} \sum_{s \in \mathcal{S}} T(s, a, s') b(s),$$

where $Pr(o|a, b) = \sum_{s, s'' \in \mathcal{S}} O(s'', a, o) T(s, a, s'') b(s)$. Using belief states, a POMDP can be rewritten as an MDP over the belief space, or *belief MDP*, $\langle \Delta, \mathcal{A}, \tau, \rho \rangle$, where the new transition $\tau$ and reward functions $\rho$ are defined respectively over $\Delta \times \mathcal{A} \times \Delta$ and $\Delta \times \mathcal{A}$. With this reformulation, a number of theoretical results about MDPs can be extended, such as the existence of a deterministic policy that is optimal. An issue is that, even if a POMDP has a finite number of states, the corresponding belief MDP is defined over a continuous—and thus infinite—belief space.

In this continuous MDP, the objective is to maximize the cumulative reward by looking for a policy taking the current belief state as input. More formally, we are searching for a policy verifying $\pi^* = \operatorname{argmax}_{\pi \in \mathcal{A}^\Delta} J^\pi(b_0)$ where $J^\pi(b_0) = E\left[\sum_{t=0}^\infty \gamma \rho_t | b_0, \pi\right]$, $\rho_t$ being the expected immediate reward obtained at time step $t$, and $\gamma$ a discount factor. Bellman's principle of optimality [7] lets us compute the function $J^{\pi^*}$ recursively through the *value function*

$$V_n(b) = \max_{a \in \mathcal{A}} \left[ \rho(b, a) + \gamma \int_{b' \in \Delta} \tau(b, a, b') V_{n-1}(b') db' \right]$$

$$= \max_{a \in \mathcal{A}} \left[ \rho(b, a) + \gamma \sum_o Pr(o|a, b) V_{n-1}(b^{a,o}) \right], \quad (1)$$

where, for all $b \in \Delta$, $V_0(b) = 0$, and $J^{\pi^*}(b) = V_{n=H}(b)$ (where $H$ is the—possibly infinite—horizon of the problem).

The POMDP framework presents a reward function $r(s, a)$ based on the state and action. On the other hand, the belief MDP presents a reward function $\rho(b, a)$ based on beliefs. This belief-based

reward function is derived as the expectation of the POMDP rewards:

$$\rho(b,a) = \sum_s b(s)r(s,a).$$ (2)

An important consequence of Equation 2 is that the recursive computation described in Eq. 1 has the property to generate piecewise-linear and convex (PWLC) value functions for each horizon [1], i.e., each function is determined by a set of hyperplanes (each represented by a vector), the value at a given belief point being that of the highest hyperplane. For example, if $\Gamma_n$ is the set of vectors representing the value function for horizon $n$, then $V_n(b) = \max_{\alpha \in \Gamma_n} \sum_s b(s)\alpha(s)$.

## 2.2 Solving POMDPs with Exact Updates

Using the PWLC property, one can perform the Bellman update using the following factorization of Eq. 1:

$$V_n(b) = \max_{a \in \mathcal{A}} \sum_o \sum_s b(s) \left[ \frac{r(s,a)}{|\Omega|} + \sum_{s'} T(s,a,s')O(s',a,o)\chi_{n-1}(b^{a,o},s') \right],$$ (3)

with[2] $\chi_n(b) = \underset{\alpha \in \Gamma_n}{\operatorname{argmax}}\, b \cdot \alpha$. If we consider the term in brackets in Eq. 3, this generates $|\Omega| \times |\mathcal{A}|$ $\Gamma$-sets, each one of size $|\Gamma_{n-1}|$. These sets are defined as

$$\overline{\Gamma}_n^{a,o} = \left\{ \frac{r^a}{|\Omega|} + P^{a,o} \cdot \alpha_{n-1} \,\middle|\, \alpha_{n-1} \in \Gamma_{n-1} \right\},$$ (4)

where $P^{a,o}(s,s') = T(s,a,s')O(s',a,o)$ and $r^a(s) = r(s,a)$. Therefore, for obtaining an exact representation of the value function, one can compute ($\bigoplus$ being the cross-sum between two sets):

$$\overline{\Gamma_n} = \bigcup_a \bigoplus_o \overline{\Gamma}_n^{a,o}.$$

Yet, these $\overline{\Gamma}_n^{a,o}$ sets—and also the final $\overline{\Gamma}_n$—are *non-parsimonious*: some $\alpha$-vectors may be useless because the corresponding hyperplanes are below the value function. Pruning phases are then required to remove dominated vectors. There are several algorithms based on pruning techniques like *Batch Enumeration* [8] or more efficient algorithms such as *Witness* or *Incremental Pruning* [6].

## 2.3 Solving POMDPs with Approximate Updates

The value function updating processes presented above are exact and provide value functions that can be used whatever the initial belief state $b_0$. A number of approximate POMDP solutions have been proposed to reduce the complexity of these computations, using for example heuristic estimates of the value function, or applying the value update only on selected belief points [9]. We focus here on the latter *point-based* (PB) approximations, which have largely contributed to the recent progress in solving POMDPs, and whose relevant literature goes from Lovejoy's early work [10] via Pineau *et al.*'s PBVI [11], Spaan and Vlassis' Perseus [12], Smith and Simmons' HSVI2 [13], through to Kurniawati *et al.*'s SARSOP [14].

At each iteration $n$ until convergence, a typical PB algorithm:

1. selects a new set of belief points $B_n$ based on $B_{n-1}$ and the current approximation $V_{n-1}$;

2. performs a Bellman backup at each belief point $b \in B_n$, resulting in one $\alpha$-vector per point;

3. prunes points whose associated hyperplanes are dominated or considered negligible.

The various PB algorithms differ mainly in how belief points are selected, and in how the update is performed. Existing belief point selection methods have exploited ideas like using a regular discretization or a random sampling of the belief simplex, picking reachable points (by simulating action sequences starting from $b_0$), adding points that reduce the approximation error, or looking in particular at regions relevant to the optimal policy [15].

# 3  POMDP extension for Active Sensing

## 3.1  Introducing $\rho$POMDPs

All problems with partial observability confront the issue of getting more information to achieve some goal. This problem is usually implicitly addressed in the resolution process, where acquiring information is only a means for optimizing an expected reward based on the system state. Some active sensing problems can be modeled this way (e.g. active classification), but not all of them. A special kind of problem is when the performance criterion incorporates an explicit measure of the agent's knowledge about the system, which is based on the beliefs rather than states. Surveillance for example is a never-ending task that does not seem to allow for a modeling with state-dependent rewards. Indeed, if we consider the simple problem of knowing the position of a hidden object, it is possible to solve this without even having seen the object (for instance if all the locations but one have been visited). However, the reward of a POMDP cannot model this since it is only based on the current state and action. One solution would be to include the whole history in the state, leading to a combinatorial explosion. We prefer to consider a new way of defining rewards based on the acquired knowledge represented by belief states. The rest of the paper explores the fact that belief MDPs can be used outside the specific definition of $\rho(b, a)$ in Eq. 2, and therefore discusses how to solve this special type of active sensing problems.

As Eq. 2 is no longer valid, the direct link with POMDPs is broken. We can however still use all the other components of POMDPs such as states, observations, etc. A way of fixing this is to generalize the POMDP framework to a $\rho$-based POMDP ($\rho$POMDP), where the reward is not defined as a function $r(s, a)$, but directly as a function $\rho(b, a)$. The nature of the $\rho(b, a)$ function depends on the problem, but is usually related to some uncertainty or error measure [3, 2, 4]. Most common methods are those based on Shannon's information theory, in particular Shannon's entropy or the Kullback-Leibler distance [16]. In order to present these functions as rewards, they have to measure information rather than uncertainty, so the negative entropy function $\rho_{ent}(b) = \log_2(|S|) + \sum_{s \in S} b(s) \log_2(b(s))$—which is maximal in the corners of the simplex and minimal in the center— is used rather than Shannon's original entropy. Also, other simpler functions based on the same idea can be used, such as the *distance from the simplex center* (DSC), $\rho_{dsc}(b) = \|b - c\|_m$, where $c$ is the center of the simplex and $m$ a positive integer that denotes the order of the metric space. Please note that $\rho(b, a)$ is not restricted to be only an uncertainty measurement, but can be a combination of the expected state-action rewards—as in Eq. 2—and an uncertainty or error measurement. For example, Mihaylova *et al.*'s work [3] defines the active sensing problem as optimizing a weighted sum of uncertainty measurements and costs, where the former depends on the belief and the latter on the system state.

In the remainder of this paper, we show how to apply classical POMDP algorithms to $\rho$POMDPs. To that end, we discuss the convexity of the value function, which permits extending these algorithms using PWLC approximations.

## 3.2  Convexity Property

An important property used to solve normal POMDPs is the result that a belief-based value function is convex, because $r(s, a)$ is linear with respect to the belief, and the expectation, sum and max operators preserve this property [1]. For $\rho$POMDPs, this property also holds if the reward function $\rho(b, a)$ is convex, as shown in Theorem 3.1.

**Theorem 3.1.** *If $\rho$ and $V_0$ are convex functions over $\Delta$, then the value function $V_n$ of the belief MDP is convex over $\Delta$ at any time step $n$. [Proof in [17, Appendix]]*

This last theorem is based on $\rho(b, a)$ being a convex function over $b$, which is a natural property for uncertainty (or information) measures, because the objective is to avoid belief distributions that do not give much information on which state the system is in, and to assign higher rewards to those beliefs that give higher probabilities of being in a specific state. Thus, a reward function meant to reduce the uncertainty must provide high payloads near the corners of the simplex, and low payloads near its center. For that reason, we will focus only on reward functions that comply with convexity in the rest of the paper.

The initial value function $V_0$ might be any convex function for infinite-horizon problems, but by

definition $V_0 = 0$ for finite-horizon problems. We will use the latter case for the rest of the paper, to provide fairly general results for both kinds of problems. Plus, starting with $V_0 = 0$, it is also easy to prove by induction that, if $\rho$ is continuous (respectively differentiable), then $V_n$ is continuous (respectively *piecewise* differentiable).

### 3.3 Piecewise Linear Reward Functions

This section focuses on the case where $\rho$ is a PWLC function and shows that only a small adaptation of the exact and approximate updates in the POMDP case is necessary to compute the optimal value function. The complex case where $\rho$ is not PWLC is left for Sec. 4.

#### 3.3.1 Exact Updates

From now on, $\rho(b, a)$, being a PWLC function, can be represented as several $\Gamma$-sets, one $\Gamma_\rho^a$ for each $a$. The reward is computed as:

$$\rho(b, a) = \max_{\alpha \in \Gamma_\rho^a} \left[ \sum_s b(s)\alpha(s) \right].$$

Using this definition leads to the following changes in Eq. 3

$$V_n(b) = \max_{a \in \mathcal{A}} \sum_s b(s) \left[ \chi_\rho^a(b, s) + \sum_o \sum_{s'} T(s, a, s')O(s', a, o)\chi_{n-1}(b^{a,o}, s') \right],$$

where $\chi_\rho^a(b, s) = \operatorname*{argmax}_{\alpha \in \Gamma_\rho^a}(b \cdot \alpha)$. This uses the $\Gamma$-set $\Gamma_\rho^a$ and generates $|\Omega| \times |A|$ $\Gamma$-sets:

$$\overline{\Gamma}_n^{a,o} = \{P^{a,o} \cdot \alpha_{n-1} | \alpha_{n-1} \in \Gamma_{n-1}\},$$

where $P^{a,o}(s, s') = T(s, a, s')O(s', a, o)$.

Exact algorithms like Value Iteration or Incremental Pruning can then be applied to this POMDP extension in a similar way as for POMDPs. The difference is that the cross-sum includes not only one $\alpha^{a,o}$ for each observation $\Gamma$-set $\overline{\Gamma}_n^{a,o}$, but also one $\alpha_\rho$ from the $\Gamma$-set $\Gamma_\rho^a$ corresponding to the reward:

$$\overline{\Gamma}_n = \bigcup_a \left[ \bigoplus_o \overline{\Gamma}_n^{a,o} \oplus \Gamma_\rho^a \right].$$

Thus, the cross-sum generates $|R|$ times more vectors than with a classic POMDP, $|R|$ being the number of $\alpha$-vectors specifying the $\rho(b, a)$ function[3].

#### 3.3.2 Approximate Updates

Point-based approximations can be applied in the same way as PBVI or SARSOP do to the original POMDP update. The only difference is again the reward function representation as an envelope of hyperplanes. PB algorithms select the hyperplane that maximizes the value function at each belief point, so the same simplification can be applied to the set $\Gamma_\rho^a$.

## 4 Generalizing to Other Reward Functions

Uncertainty measurements such as the negative entropy or the DSC (with $m > 1$ and $m \neq \infty$) are not piecewise linear functions. In theory, each step of value iteration can be analytically computed using these functions, but the expressions are not closed as in the linear case, growing in complexity and making them unmanageable after a few steps. Moreover, pruning techniques cannot be applied directly to the resulting hypersurfaces, and even second order measures do not exhibit standard quadratic forms to apply quadratic programming. However, convex functions can be efficiently approximated by piecewise linear functions, making it possible to apply the techniques described in Section 3.3 with a bounded error, as long as the approximation of $\rho$ is bounded.

### 4.1 Approximating $\rho$

Consider a continuous, convex and piecewise differentiable reward function $\rho(b)$,[4] and an arbitrary (and finite) set of points $B \subset \Delta$ where the gradient is well defined. A lower PWLC approximation of $\rho(b)$ can be obtained by using each element $b' \in B$ as a base point for constructing a tangent hyperplane which is always a lower bound of $\rho(b)$. Concretely, $\omega_{b'}(b) = \rho(b') + (b - b') \cdot \nabla\rho(b')$ is the linear function that represents the tangent hyperplane. Then, the approximation of $\rho(b)$ using a set $B$ is defined as $\omega_B(b) = \max_{b'}(\omega_{b'}(b))$.

At any point $b \in \Delta$ the error of the approximation can be written as

$$\epsilon_B(b) = |\rho(b) - \omega_B(b)|, \tag{5}$$

and if we specifically pick $b$ as the point where $\epsilon_B(b)$ is maximal (worst error), then we can try to bound this error depending on the nature of $\rho$.

It is well known that a piecewise linear approximation of a Lipschitz function is bounded because the gradient $\nabla\rho(b')$ that is used to construct the hyperplane $\omega_{b'}(b)$ has bounded norm [18]. Unfortunately, the negative entropy is not Lipschitz ($f(x) = x \log_2(x)$ has an infinite slope when $x \to 0$), so this result is not generic enough to cover a wide range of active sensing problems. Yet, under certain mild assumptions a proper error bound can still be found.

The aim of the rest of this section is to find an error bound in three steps. First, we will introduce some basic results over the simplex and the convexity of $\rho$. Informally, Lemma 4.1 will show that, for each $b$, it is possible to find a belief point in $B$ far enough from the boundary of the simplex but within a bounded distance to $b$. Then, in a second step, we will assume the function $\rho(b)$ verifies the $\alpha$-Hölder condition to be able to bound the norm of the gradient in Lemma 4.2. In the end, Theorem 4.3 will use both lemmas to bound the error of $\rho$'s approximation under these assumptions.

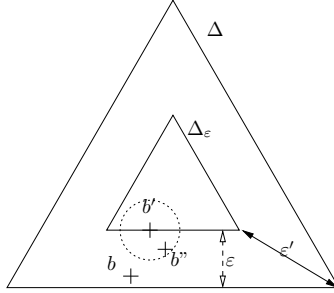

Figure 1: Simplices $\Delta$ and $\Delta_\varepsilon$, and the points $b$, $b'$ and $b''$.

For each point $b \in \Delta$, it is possible to associate a point $b^* = \mathrm{argmax}_{x \in B}\, \omega_x(b)$ corresponding to the point in $B$ whose tangent hyperplane gives the best approximation of $\rho$ at $b$. Consider the point $b \in \Delta$ where $\epsilon_B(b)$ is maximum: this error can be easily computed using the gradient $\nabla\rho(b^*)$. Unfortunately, some partial derivatives of $\rho$ may diverge to infinity on the boundary of the simplex in the non-Lipschitz case, making the error hard to analyze. Therefore, to ensure that this error can be bounded, instead of $b^*$, we will take a safe $b'' \in B$ (far enough from the boundary) by using an intermediate point $b'$ in an *inner simplex* $\Delta_\varepsilon$, where $\Delta_\varepsilon = \{b \in [\varepsilon, 1]^{\mathcal{N}} \mid \sum_i b_i = 1\}$ with $\mathcal{N} = |\mathcal{S}|$.

Thus, for a given $b \in \Delta$ and $\varepsilon \in (0, \frac{1}{\mathcal{N}}]$, we define the point $b' = \mathrm{argmin}_{x \in \Delta_\varepsilon} \|x - b\|_1$ as the closest point to $b$ in $\Delta_\varepsilon$ and $b'' = \mathrm{argmin}_{x \in B} \|x - b'\|_1$ as the closest point to $b'$ in $B$ (see Figure 1). These two points will be used to find an upper bound for the distance $\|b - b''\|_1$ based on the *density* of $B$, defined as $\delta_B = \min_{b \in \Delta} \max_{b' \in B} \|b - b'\|_1$.

**Lemma 4.1.** *The distance (1-norm) between the maximum error point $b \in \Delta$ and the selected $b'' \in B$ is bounded by $\|b - b''\|_1 \leq 2(\mathcal{N} - 1)\varepsilon + \delta_B$. [Proof in [17, Appendix]]*

If we pick $\varepsilon > \delta_B$, then we are sure that $b''$ is not on the boundary of the simplex $\Delta$, with a minimum distance from the boundary of $\eta = \varepsilon - \delta_B$. This will allow finding bounds for the PWLC

approximation of convex $\alpha$-Hölder functions, which is a broader family of functions including the negative entropy, convex Lipschitz functions and others. The $\alpha$-Hölder condition is a generalization of the Lipschitz condition. In our setting it means, for a function $f : \mathcal{D} \mapsto \mathbb{R}$ with $\mathcal{D} \subset \mathbb{R}^n$, that it complies with

$$\exists \alpha \in (0, 1], \, \exists K_\alpha > 0, \text{ s.t. } |f(x) - f(y)| \leq K_\alpha \|x - y\|_1^\alpha.$$

The limit case, where a convex $\alpha$-Hölder function has infinite-valued norm for the gradient, is always on the boundary of the simplex $\Delta$ (due to the convexity), and therefore the point $b''$ will be free of this predicament because of $\eta$. More precisely, an $\alpha$-Hölder function in $\Delta$ with constant $K_\alpha$ in 1-norm complies with the Lipschitz condition on $\Delta_\eta$ with a constant $K_\alpha \eta^\alpha$ (see [17, Appendix]). Moreover, the norm of the gradient $\|\nabla f(b'')\|_1$ is also bounded as stated by Lemma 4.2.

**Lemma 4.2.** *Let $\eta > 0$ and $f$ be an $\alpha$-Hölder (with constant $K_\alpha$), bounded and convex function from $\Delta$ to $\mathbb{R}$, $f$ being differentiable everywhere in $\Delta^o$ (the interior of $\Delta$). Then, for all $b \in \Delta_\eta$, $\|\nabla f(b)\|_1 \leq K_\alpha \eta^{\alpha-1}$. [Proof in [17, Appendix]]*

Under these conditions, we can show that the PWLC approximation is bounded.

**Theorem 4.3.** *Let $\rho$ be a continuous and convex function over $\Delta$, differentiable everywhere in $\Delta^o$ (the interior of $\Delta$), and satisfying the $\alpha$-Hölder condition with constant $K_\alpha$. The error of an approximation $\omega_B$ can be bounded by $C\delta_b^\alpha$, where $C$ is a scalar constant. [Proof in [17, Appendix]]*

## 4.2 Exact Updates

Knowing that the approximation of $\rho$ is bounded for a wide family of functions, the techniques described in Sec. 3.3.1 can be directly applied using $\omega_B(b)$ as the PWLC reward function. These algorithms can be safely used because the propagation of the error due to exact updates is bounded. This can be proven using a similar methodology as in [11, 10]. Let $V_t$ be the value function using the PWLC approximation described above and $V_t^*$ the optimal value function both at time $t$, $H$ being the exact update operator and $\hat{H}$ the same operator with the PWLC approximation. Then, the error from the real value function is

$$\begin{aligned}
\|V_t - V_t^*\|_\infty &= \|\hat{H}V_{t-1} - HV_{t-1}^*\|_\infty & \text{(By definition)} \\
&\leq \|\hat{H}V_{t-1} - HV_{t-1}\|_\infty + \|HV_{t-1} - HV_{t-1}^*\|_\infty & \text{(By triangular inequality)} \\
&\leq |\omega_{b^*} + \alpha_{b^*} \cdot b - \rho(b) - \alpha_{b^*} \cdot b| + \|HV_{t-1} - HV_{t-1}^*\|_\infty & \text{(Maximum error at } b) \\
&\leq C\delta_B^\alpha + \|HV_{t-1} - HV_{t-1}^*\|_\infty & \text{(By Theorem 4.3)} \\
&\leq C\delta_B^\alpha + \gamma\|V_{t-1} - V_{t-1}^*\| & \text{(By contraction)} \\
&\leq \frac{C\delta_B^\alpha}{1 - \gamma} & \text{(By sum of a geometric series)}
\end{aligned}$$

For these algorithms, the selection of the set $B$ remains open, raising similar issues as the selection of belief points in PB algorithms.

## 4.3 Approximate Updates

In the case of PB algorithms, the extension is also straightforward, and the algorithms described in Sec. 3.3.2 can be used with a bounded error. The selection of $B$, the set of points for the PWLC approximation, and the set of points for the algorithm, can be shared[5]. This simplifies the study of the bound when using both approximation techniques at the same time. Let $\hat{V}_t$ be the value function at time $t$ calculated using the PWLC approximation and a PB algorithm. Then the error between $\hat{V}_t$ and $V_t^*$ is $\|\hat{V}_t - V_t^*\|_\infty \leq \|\hat{V}_t - V_t\|_\infty + \|V_t - V_t^*\|_\infty$. The second term is the same as in Sec. 4.2, so it is bounded by $\frac{C\delta_B^\alpha}{1-\gamma}$. The first term can be bounded by the same reasoning as in [11], where $\|\hat{V}_t - V_t\|_\infty = \frac{(R_{max} - R_{min} + C\delta_B^\alpha)\delta_B}{1-\gamma}$, with $R_{min}$ and $R_{max}$ the minimum and maximum values for

$\rho(b)$ respectively. This is because the worst case for an $\alpha$ vector is $\frac{R_{min}-\epsilon}{1-\gamma}$, meanwhile the best case is only $\frac{R_{max}}{1-\gamma}$ because the approximation is always a lower bound.

## 5  Conclusions

We have introduced $\rho$POMDPs, an extension of POMDPs that allows for expressing sequential decision-making problems where reducing the uncertainty on some state variables is an explicit objective. In this model, the reward $\rho$ is typically a convex function of the belief state.

Using the convexity of $\rho$, a first important result that we prove is that a Bellman backup $V_n = HV_{n-1}$ preserves convexity. In particular, if $\rho$ is PWLC and the value function $V_0$ is equal to 0, then $V_n$ is also PWLC and it is straightforward to adapt many state-of-the-art POMDP algorithms. Yet, if $\rho$ is not PWLC, performing exact updates is much more complex. We therefore propose employing PWLC approximations of the convex reward function at hand to come back to a simple case, and show that the resulting algorithms converge to the optimal value function in the limit.

Previous work has already introduced belief-dependent rewards, such as Spaan's discussion about POMDPs and Active Perception [19], or Hero et al.'s work in sensor management using POMDPs [5]. Yet, the first one only presents the problem of non-PWLC value functions without giving a specific solution, meanwhile the second solves the problem using Monte-Carlo techniques that do not rely on the PWLC property. In the robotics field, uncertainty measurements within POMDPs have been widely used as heuristics [2], with very good results but no convergence guarantees. These techniques use only state-dependent rewards, but uncertainty measurements are employed to speed up the solving process, at the cost of losing some basic properties (e.g. Markovian property). Our work paves the way for solving problems with belief-dependent rewards, using new algorithms approximating the value function (e.g. point-based ones) in a theoretically sound manner.

An important point is that the time complexity of the new algorithms only changes due to the size of the approximation of $\rho$. Future work includes conducting experiments to measure the increase in complexity. A more complex task is to evaluate the quality of the resulting approximations due to the lack of other algorithms for $\rho$POMDPs. An option is to look at online Monte-Carlo algorithms [20] as they should require little changes.

## Acknowledgements

This research was supported by the *CONICYT-Embassade de France* doctoral grant and the CO-MAC project. We would also like to thank Bruno Scherrer for the insightful discussions and the anonymous reviewers for their helpful comments and suggestions.

## Footnotes

[1] $\Pi(\mathcal{S})$ forms a simplex because $\|b\|_1 = 1$, that is why we use $\Delta$ as the set of all possible $b$.

[2]The $\chi$ function returns a vector, so $\chi_n(b,s) = (\chi_n(b))(s)$.

[3]More precisely, the number $|R|$ depends on the considered action.

[4]For convenience—and without loss of generality—we only consider the case where $\rho(b, a) = \rho(b)$.

[5]Points from $\Delta$'s boundary can be removed where the gradient is not defined, as the proofs only rely on interior points.

## References

[1] R. Smallwood and E. Sondik. The optimal control of partially observable Markov decision processes over a finite horizon. *Operation Research*, 21:1071–1088, 1973.

[2] S. Thrun. Probabilistic algorithms in robotics. *AI Magazine*, 21(4):93–109, 2000.

[3] L. Mihaylova, T. Lefebvre, H. Bruyninckx, K. Gadeyne, and J. De Schutter. Active sensing for robotics - a survey. In *Proc. 5th Intl. Conf. On Numerical Methods and Applications*, 2002.

[4] S. Ji and L. Carin. Cost-sensitive feature acquisition and classification. *Pattern Recogn.*, 40(5):1474–1485, 2007.

[5] A. Hero, D. Castan, D. Cochran, and K. Kastella. *Foundations and Applications of Sensor Management*. Springer Publishing Company, Incorporated, 2007.

[6] A. Cassandra. *Exact and approximate algorithms for partially observable Markov decision processes*. PhD thesis, Providence, RI, USA, 1998.

[7] R. Bellman. The theory of dynamic programming. *Bull. Amer. Math. Soc.*, 60:503–516, 1954.

[8] G. Monahan. A survey of partially observable Markov decision processes. *Management Science*, 28:1–16, 1982.

[9] M. Hauskrecht. Value-function approximations for partially observable Markov decision processes. *Journal of Artificial Intelligence Research*, 13:33–94.

[10] W. Lovejoy. Computationally feasible bounds for partially observed Markov decision processes. *Operations Research*, 39(1):162–175.

[11] J. Pineau, G. Gordon, and S. Thrun. Anytime point-based approximations for large POMDPs. *Journal of Artificial Intelligence Research (JAIR)*, 27:335–380, 2006.

[12] M. Spaan and N. Vlassis. Perseus: Randomized point-based value iteration for POMDPs. *Journal of Artificial Intelligence Research*, 24:195–220, 2005.

[13] T. Smith and R. Simmons. Point-based POMDP algorithms: Improved analysis and implementation. In *Proc. of the Int. Conf. on Uncertainty in Artificial Intelligence (UAI)*, 2005.

[14] H. Kurniawati, D. Hsu, and W. Lee. SARSOP: Efficient point-based POMDP planning by approximating optimally reachable belief spaces. In *Robotics: Science and Systems IV*, 2008.

[15] R. Kaplow. Point-based POMDP solvers: Survey and comparative analysis. Master's thesis, Montreal, Quebec, Canada, 2010.

[16] T. Cover and J. Thomas. *Elements of Information Theory*. Wiley-Interscience, 1991.

[17] M. Araya-López, O. Buffet, V. Thomas, and F. Charpillet. A POMDP extension with belief-dependent rewards – extended version. Technical Report RR-7433, INRIA, Oct 2010. (See also NIPS supplementary material).

[18] R. Saigal. On piecewise linear approximations to smooth mappings. *Mathematics of Operations Research*, 4(2):153–161, 1979.

[19] M. Spaan. Cooperative active perception using POMDPs. In *AAAI 2008 Workshop on Advancements in POMDP Solvers*, July 2008.

[20] S. Ross, J. Pineau, S. Paquet, and B. Chaib-draa. Online planning algorithms for POMDPs. *Journal of Artificial Intelligence Research (JAIR)*, 32:663–704, 2008.

